# Noise Thresholds for Spectral Clustering

**Sivaraman Balakrishnan**      **Min Xu**      **Akshay Krishnamurthy**      **Aarti Singh**

**School of Computer Science, Carnegie Mellon University**
`{sbalakri,minx,akshaykr,aarti}@cs.cmu.edu`

## Abstract

Although spectral clustering has enjoyed considerable empirical success in machine learning, its theoretical properties are not yet fully developed. We analyze the performance of a spectral algorithm for hierarchical clustering and show that on a class of hierarchically structured similarity matrices, this algorithm can tolerate noise that grows with the number of data points while still perfectly recovering the hierarchical clusters with high probability. We additionally improve upon previous results for $k$-way spectral clustering to derive conditions under which spectral clustering makes no mistakes. Further, using minimax analysis, we derive tight upper and lower bounds for the clustering problem and compare the performance of spectral clustering to these information theoretic limits. We also present experiments on simulated and real world data illustrating our results.

## 1   Introduction

*Clustering*, a fundamental and ubiquitous problem in machine learning, is the task of organizing data points into homogenous groups using a given measure of similarity. Two popular forms of clustering are $k$-*way*, where an algorithm directly partitions the data into $k$ disjoint sets, and *hierarchical*, where the algorithm organizes the data into a hierarchy of groups. Popular algorithms for the $k$-way problem include $k$-means, spectral clustering, and density-based clustering, while *agglomerative* methods that merge clusters from the bottom up are popular for the latter problem.

Spectral clustering algorithms embed the data points by projection onto a few eigenvectors of (some form of) the graph Laplacian matrix and use this spectral embedding to find a clustering. This technique has been shown to work on various arbitrarily shaped clusters and, in addition to being straightforward to implement, often outperforms traditional clustering algorithms such as the k-means algorithm.

Real world data is inevitably corrupted by noise and it is of interest to study the robustness of spectral clustering algorithms. This is the focus of our paper.

Our main contributions are:

- We leverage results from perturbation theory in a novel analysis of a spectral algorithm for hierarchical clustering to understand its behavior in the presence of noise. We provide strong guarantees on its correctness; in particular, we show that the amount of noise spectral clustering tolerates can grow rapidly with the size of the smallest cluster we want to resolve.
- We sharpen existing results on $k$-way spectral clustering. In contrast with earlier work, we provide precise error bounds through a careful characterization of a $k$-means style algorithm run on the spectral embedding of the data.
- We also address the issue of optimal noise thresholds via the use of minimax theory. In particular, we establish tight information-theoretic upper and lower bounds for cluster resolvability.

## 2 Related Work and Definitions

There are several high-level justifications for the success of spectral clustering. The algorithm has deep connections to various graph-cut problems, random walks on graphs, electric network theory, and via the graph Laplacian to the Laplace-Beltrami operator. See [16] for an overview.

Several authors (see von Luxburg et. al. [17] and references therein) have shown various forms of asymptotic convergence for the Laplacian of a graph constructed from random samples drawn from a distribution on or near a manifold. These results however often do not easily translate into precise guarantees for successful recovery of clusters, which is the emphasis of our work.

There has also been some theoretical work on spectral algorithms for cluster recovery in random graph models. McSherry [9] studies the "cluster-structured" random graph model in which the probability of adding an edge can vary depending on the clusters the edge connects. He considers a specialization of this model, the planted partition model, which specifies only two probabilities, one for inter-cluster edges and another for intra-cluster edges. In this case, we can view the observed adjacency matrix as a random perturbation of a low rank "expected" adjacency matrix which encodes the cluster membership. McSherry shows that one can recover the clusters from a low rank approximation of the observed (noisy) adjacency matrix. These results show that low-rank matrices have spectra that are robust to noise. Our results however, show that we can obtain similar insensitivity (to noise) guarantees for a class of interesting structured *full-rank* matrices, indicating that this robustness extends to a much broader class of matrices.

More recently, Rohe et al [11] analyze spectral clustering in the stochastic block model (SBM), which is an example of a structured random graph. They consider the *high-dimensional* scenario where the number of clusters $k$ grows with the number of data points $n$ and show that under certain assumptions the *average* number of mistakes made by spectral clustering $\rightarrow 0$ with increasing $n$. Our work on hierarchical clustering also has the same high-dimensional flavor since the number of clusters we resolve grows with $n$. However, in the hierarchical clustering setting, errors made at the bottom level propogate up the tree and we need to make precise arguments to ensure that the *total* number of errors $\rightarrow 0$ with increasing $n$ (see Theorem 1).

Since Rohe et al [11] and McSherry [9] consider random graph models, the "noise" on each entry has *bounded* variance. We consider more general noise models and study the relation between errors in clustering and noise variance. Another related line of work is on the problem of spectrally separating mixtures of Gaussians [1, 2, 8].

Ng et al. [10] study k-way clustering and show that the eigenvectors of the graph Laplacian are stable in 2-norm under small perturbations. This justifies the use of $k$-means in the perturbed subspace since ideally without noise, the spectral embedding by the top $k$ eigenvectors of the graph Laplacian reflects the true cluster memberships, However, closeness in 2-norm does not translate into a strong bound on the *total number* of errors made by spectral clustering.

Huang et al. [7] study the misclustering rate of spectral clustering under the somewhat unnatural assumption that every coordinate of the Laplacian's eigenvectors are perturbed by independent and identically distributed noise. In contrast, we specify our noise model as an additive perturbation to the similarity matrix, making no direct assumptions on how this affects the spectrum of the Laplacian. We show that the eigenvectors are stable in $\infty$-norm and use this result to precisely bound the misclustering rate of our algorithm.

### 2.1 Definitions

The clustering problem can be defined as follows: Given an $(n \times n)$ similarity matrix on $n$ data points, find a set $\mathcal{C}$ of subsets of the points such that points belonging to the same subset have high similarity and points in different subsets have low similarity. Our first results focus on *binary* hierarchical clustering, which is formally defined as follows:

**Definition 1** *A **hierarchical clustering** $\mathcal{T}$ on data points $\{X_i\}_{i=1}^n$ is a collection of clusters (subsets of the points) such that $C_0 := \{X_i\}_{i=1}^n \in \mathcal{T}$ and for any $C_i, C_j \in \mathcal{T}$, either $C_i \subset C_j$, $C_j \subset C_i$, or $C_i \cap C_j = \emptyset$. A **binary hierarchical clustering** $\mathcal{T}$ is a hierarchical clustering such that for each non-atomic $C_k \in \mathcal{T}$, there exists two proper subsets $C_i, C_j \in \mathcal{T}$ with $C_i \cap C_j = \emptyset$ and $C_i \cup C_j = C_k$. We label each cluster by a sequence $s$ of Ls and Rs so that $C_{s \cdot L}$ and $C_{s \cdot R}$ partitions $C_s$, $C_{s \cdot LL}$ and $C_{s \cdot LR}$ partititons $C_{s \cdot L}$, and so on.*

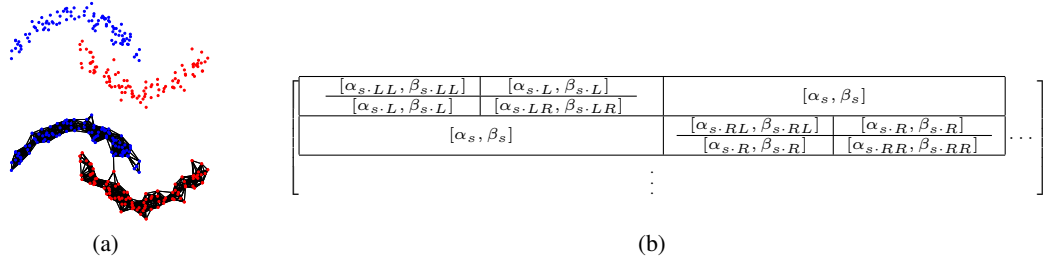

(a)                                                    (b)

Figure 1: (a): Two moons data set (Top). For a similarity function defined on the $\epsilon$-neighborhood graph (Bottom), this data set forms an ideal matrix. (b) An ideal matrix for the hierarchical problem.

Ideally, we would like that at all levels of the hierarchy, points within a cluster are more similar to each other than to points outside of the cluster. For a suitably chosen similarity function, a data set consisting of clusters that lie on arbitrary manifolds with complex shapes can result in this ideal case. As an example, in the two-moons data set in Figure 1(a), the popular technique of constructing a nearest neighbor graph and defining the distance between two points as the length of the *longest* edge on the *shortest* path between them results in an ideal similarity matrix. Other non-Euclidean similarity metrics (for instance density based similarity metrics [12]) can also allow for non-parametric cluster shapes.

For such ideal similarity matrices, we can show that the spectral clustering algorithm will deterministically recover all clusters in the hierarchy (see Theorem 5 in the appendix). However, since this ideal case does not hold in general, we focus on similarity matrices that can be decomposed into an ideal matrix and a high-variance noise term.

**Definition 2** *A similarity matrix $W$ is a **noisy hierarchical block matrix** (noisy HBM) if $W \triangleq A + R$ where $A$ is ideal and $R$ is a perturbation matrix, defined as follows:*

- *An **ideal similarity matrix**, shown in Figure 1(b), is characterized by ranges of off-block-diagonal similarity values $[\alpha_s, \beta_s]$ for each cluster $C_s$ such that if $x \in C_{s\cdot L}$ and $y \in C_{s\cdot R}$ then $\alpha_s \leq A_{xy} \leq \beta_s$. Additionally, $\min\{\alpha_{s\cdot R}, \alpha_{s\cdot L}\} > \beta_s$.*
- *A symmetric $(n \times n)$ matrix $R$ is a **perturbation matrix** with parameter $\sigma$ if (a) $\mathbb{E}(R_{ij}) = 0$, (b) the entries of $R$ are subgaussian, that is $\mathbb{E}(\exp(tR_{ij})) \leq \exp(\frac{\sigma^2 t^2}{2})$ and (c) for each row $i$, $R_{i1}, \ldots, R_{in}$ are independent.*

The perturbations we consider are quite general and can accommodate bounded (with $\sigma$ upper bounded by the range), Gaussian (where $\sigma$ is the standard deviation), and several other common distributions. This model is well-suited to noise that arises from the direct measurement of similarities. It is also possible to assume instead that the measurements of individual data points are noisy though we do not focus on this case in our paper.

In the $k$-way case, we consider the following similarity matrix which is studied by Ng et. al [10].

**Definition 3** *$W$ is a **noisy $k$-Block Diagonal** matrix if $W \triangleq A + R$ where $R$ is a perturbation matrix and $A$ is an ideal matrix for the $k$-way problem. An ideal matrix for the $k$-way problem has within-cluster similarities larger than $\beta_0 > 0$ and between cluster similarities 0.*

Finally, we define the combinatorial Laplacian matrix, which will be the focus of our spectral algorithm and our subsequent analysis.

**Definition 4** *The **combinatorial Laplacian** $L$ of a matrix $W$ is defined as $L \triangleq D - W$ where $D$ is a diagonal matrix with $D_{ii} \triangleq \sum_{j=1}^{n} W_{ij}$.*

We note that other analyses of spectral clustering have studied other Laplacian matrices, particularly, the *normalized Laplacians* defined as $L_n \triangleq D^{-1}L$ and $L_n \triangleq D^{-\frac{1}{2}}LD^{-\frac{1}{2}}$. However as we show in Appendix E, the normalized Laplacian can mis-cluster points even for an ideal noiseless similarity matrix.

**Algorithm 1** HS

___
**input** (noisy) $n \times n$ similarity matrix $W$
   Compute Laplacian $L = D - W$
   $v_2 \leftarrow$ smallest non-constant eigenvector of $L$
   $C_1 \leftarrow \{i : v_2(i) \geq 0\}, C_2 \leftarrow \{j : v_2(j) < 0\}$
   $\mathcal{C} \leftarrow \{C_1, C_2\} \cup$ HS $(W_{C_1}) \cup$ HS $(W_{C_2})$
**output** $\mathcal{C}$
___

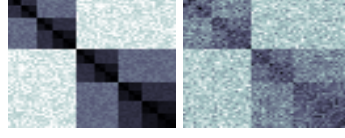

Figure 2: An ideal matrix and a noisy HBM. Clusters at finer granularity are masked by noise.

**Algorithm 2** K-WAY SPECTRAL

___
**input** (noisy) $n \times n$ similarity matrix $W$, number of clusters $k$
   Compute Laplacian $L = D - W$
   $V \leftarrow (n \times k)$ matrix with columns $v_1, ..., v_k$, where $v_i \triangleq i$th smallest eigenvector of $L$
   $c_1 \leftarrow V_1$ (the first row of $V$).
   For $i = 2 \ldots k$ let $c_i \leftarrow \mathrm{argmax}_{j \in \{1...n\}} \min_{l \in \{1,...,i-1\}} ||V_j - V_{c_l}||_2$.
   For $i = 1 \ldots n$ set $c(i) = \mathrm{argmin}_{j \in \{1...k\}} ||V_i - V_{c_j}||_2$
**output** $\mathcal{C} \triangleq \{\{j \in \{1 \ldots n\} : c(j) = i\}\}_{i=1}^k$
___

# 3 Algorithms and Main Results

In our analysis we study the algorithms for hierarchical and $k$-way clustering, outlined in Algorithms 1 and 2. Both of these algorithms take a similarity matrix $W$ and compute the eigenvectors corresponding to the smallest eigenvalues of the Laplacian of $W$. The algorithms then run simple procedures to recover the clustering from the spectral embedding of the data points by these eigenvectors. Our Algorithm 2 deviates slightly from the standard practice of running $k$-means in the perturbed subspace. We instead use the optimal algorithm for the $k$-center problem (Hochbaum-Shmoys [6]) because of its amenability to theoretical analysis. We will in this section outline our main results; we sketch the proofs in the next section and defer full proofs to the Appendix.

We first state the following general assumptions, which we place on the *ideal* similarity matrix $A$:

**Assumption 1** *For all $i, j$, $0 < A_{ij} \leq \beta^*$ for some constant $\beta^*$.*

**Assumption 2** *(Balanced clusters) There is a constant $\eta \geq 1$ such that at every split of the hierarchy $\frac{|C_{\max}|}{|C_{\min}|} \leq \eta$, where $|C_{\max}|, |C_{\min}|$ are the sizes of the biggest and smallest clusters respectively.*

**Assumption 3** *(Range Restriction) For every cluster $s$, $\min\{\alpha_{s \cdot L}, \alpha_{s \cdot R}\} - \beta_s > \eta(\beta_s - \alpha_s)$.*

It is important to note that these assumptions are placed *only* on the ideal matrices. The noisy HBMs can with high probability violate these assumptions.

We assume that the entries of $A$ are strictly greater than 0 for technical reasons; we believe, as confirmed empirically, that this restriction is not necessary for our results to hold. Assumption 2 says that at every level the largest cluster is only a constant fraction larger than the smallest. This can be relaxed albeit at the cost of a worse rate. For the ideal matrix, the Assumption 3 ensures that at every level of the hierarchy, the gap between the within-cluster similarities and between-cluster similarities is larger than the range of between-cluster similarities. Earlier papers [9, 11] assume that the ideal similarities are constant within a block in which case the assumption is trivially satisfied by the definition of the ideal matrix. However, more generally this assumption is necessary to show that the entries of the eigenvector are safely bounded away from zero. If this assumption is violated by the ideal matrix, then the eigenvector entries can decay as fast as $O(1/n)$ (see Appendix E for more details), and our analysis shows that such matrices will no longer be robust to noise.

Other analyses of spectral clustering often directly make less interpretable assumptions about the spectrum. For instance, Ng et al. [10] assume conditions on the eigengap of the normalized Laplacian and this assumption implicitly creates constraints on the entries of the ideal matrix $A$ that can be hard to make explicit.

To state our theorems concisely we will define an additional quantity $\gamma_{\mathcal{S}}^*$. Intuitively, $\gamma_{\mathcal{S}}^*$ quantifies how close the ideal matrix comes to violating Assumption 3 over a set of clusters $\mathcal{S}$.

**Definition 5** *For a set of clusters* $\mathcal{S}$, *define* $\gamma_{\mathcal{S}}^* \triangleq \min_{s \in \mathcal{S}} \min\{\alpha_{s \cdot L}, \alpha_{s \cdot R}\} - \beta_s - \eta(\beta_s - \alpha_s)$.

We, as well as previous works [10, 11], rely on results from perturbation theory to bound the error in the observed eigenvectors in 2-norm. Using this approach, the straightforward way to analyze the number of errors is pessimistic since it assumes the difference between the two eigenvectors is concentrated on a few entries. However, we show that the perturbation is in fact generated by a random process and thus unlikely to be adversarially concentrated. We formalize this intuition to *uniformly* bound the perturbations on every entry and get a stronger guarantee.

We are now ready to state our main result for hierarchical spectral clustering. At a high level, this result gives conditions on the noise scale factor $\sigma$ under which Algorithm HS will recover all clusters $s \in \mathcal{S}_m$, where $\mathcal{S}_m$ is the set of all clusters of size at least $m$.

**Theorem 1** *Suppose that* $W = A + R$ *is an* $(n \times n)$ *noisy HBM where $A$ satisfies Assumptions 1, 2, and 3. Suppose that the scale factor of $R$ increases at* $\sigma = o\left(\min\left(\kappa^{\star 5}\sqrt{\frac{m}{\log n}}, \kappa^{\star 4}\sqrt[4]{\frac{m}{\log n}}\right)\right)$ *where* $\kappa^\star = \min\left(\alpha_0, \frac{\gamma_{\mathcal{S}m}^\star}{1+\eta}\right)$, $m > 0$ *and* $m = \omega(\log n)$ [1]. *Then for all $n$ large enough, with probability at least* $1 - 6/n$, HS , *on input M, will* exactly *recover all clusters of size at least $m$.*

A few remarks are in order:
1. It is impossible to resolve the entire hierarchy, since small clusters can be irrecoverably buried in noise. The amount of noise that algorithm HS can tolerate is directly dependent on the size of the smallest cluster we want to resolve.
2. As a consequence of our proof, we show that to resolve only the first level of the hierarchy, the amount of noise we can tolerate is (pessimistically) $o(\kappa^{\star 5}\sqrt[4]{n/\log n})$ which grows rapidly with $n$.
3. Under this scaling between $n$ and $\sigma$, it can be shown that popular agglomerative algorithms such as single linkage will fail with high probability. We verify this negative result through experiments (see Section 5).
4. Since we assume that $\beta^*$ does not grow with $n$, both the range $(\beta_s - \alpha_s)$ and the gap $(\min\{\alpha_{s \cdot L}, \alpha_{s \cdot R}\} - \beta_s)$ must decrease with $n$ and hence that $\gamma_{\mathcal{S}_m}^*$ must decrease as well. For example, if we have uniform ranges and gaps across all levels, then $\gamma_{\mathcal{S}_m}^* = \Theta(1/\log n)$.

   For constant $\alpha_0$, for $n$ large enough $\kappa^\star = \frac{\gamma_{\mathcal{S}m}^\star}{1+\eta}$. We see that in our analysis $\gamma_{\mathcal{S}m}^\star$ is a crucial determinant of the noise tolerance of spectral clustering.

We extend the intuition behind Theorem 1 to the $k$-way setting. Some arguments are more subtle since spectral clustering uses the *subspace* spanned by the $k$ smallest eigenvectors of the Laplacian. We improve the results of Ng et. al. [10] to provide a coordinate-wise bound on the perturbation of the subspace, and use this to make precise guarantees for Algorithm K-WAY SPECTRAL.

**Theorem 2** *Suppose that* $W = A + R$ *is an* $(n \times n)$ ***noisy k-Block Diagonal*** *matrix where $A$ satisfies Assumptions 1 and 2. Suppose that the scale factor of $R$ increases at rate* $\sigma = o(\frac{\beta_0}{k}(\frac{n}{k \log n})^{1/4})$. *Then with probability* $1 - 8/n$, *for all $n$ large enough,* K-WAY SPECTRAL *will exactly recover the $k$ clusters.*

### 3.1 Information-Theoretic Limits

Having introduced our analysis for spectral clustering a pertinent question remains. *Is the algorithm optimal in its dependence on the various parameters of the problem?*

We establish the minimax rate in the simplest setting of a single binary split and compare it to our own results on spectral clustering. With the necessary machinery in place, the minimax rate for the $k$-way problem follows easily. We derive lower bounds on the problem of correctly identifying two clusters under the assumption that the clusters are balanced. In particular, we derive conditions on $(n, \sigma, \gamma)$, i.e. the number of objects, the noise variance and the gap between inter and intra-cluster similarities, under which *any* method will make an error in identifying the correct clusters.

**Theorem 3** *There exists a constant $\alpha \in (0, 1/8)$ such that if, $\sigma \geq \gamma \sqrt{\frac{n}{\alpha \log\left(\frac{n}{2}\right)}}$ the probability of failure of any estimator of the clustering remains bounded away from 0 as $n \to \infty$.*

Under the conditions of this Theorem $\gamma$ and $\kappa^\star$ coincide, provided the inter-cluster similarities remain bounded away from 0 by at least a constant. As a direct consequence of Theorem 1, spectral clustering requires $\sigma \leq \min\left(\gamma^5 \sqrt{\frac{n}{C \log\left(\frac{n}{2}\right)}}, \gamma^4 \sqrt[4]{\frac{n}{C \log\left(\frac{n}{2}\right)}}\right)$ (for a large enough constant $C$).

Thus, the noise threshold for spectral clustering does not match the lower bound. To establish that this lower bound is indeed tight, we need to demonstrate a (not necessarily computationally efficient) procedure that achieves this rate. We analyze a combinatorial procedure that solves the NP-hard problem of finding the minimum cut of size exactly $n/2$ by searching over all subsets. This algorithm is strongly related to spectral clustering with the combinatorial Laplacian, which solves a *relaxation* of the balanced minimum cut problem. We prove the following theorem in the appendix.

**Theorem 4** *There exists a constant $C$ such that if $\sigma < \gamma \sqrt{\frac{n}{C \log\left(\frac{n}{2}\right)}}$ the combinatorial procedure described above succeeds with probability at least $1 - \frac{1}{n}$ which goes to 0 as $n \to \infty$.*

This theorem and the lower bound together establish the minimax rate. It however, remains an open problem to tighten the analysis of spectral clustering in this paper to match this rate. In the Appendix we modify the analysis of [9] to show that under the added restriction of block constant ideal similarities there is an efficient algorithm that achieves the minimax rate.

## 4  Proof Outlines

Here, we present proof sketches of our main theorems, deferring the details to the Appendix.

**Outline of proof of Theorem 1**

Let us first restrict our attention toward finding the first split in the hierarchical clustering. Once we prove that we can recover the first split correctly, we can then recursively apply the same arguments along with some delicate union bounds to prove that we will recover all large-enough splits of the hierarchy. To make presentation clearer, we will only focus here on the scaling between $\sigma^2$ and $n$. Of course, when we analyze deeper splits, $n$ becomes the size of the sub-cluster.

Let $W = A + R$ be the $n \times n$ noisy HBM. One can readily verify that the Laplacian of $W$, $L_W$, can be decomposed as $L_A + L_R$. Let $v^{(2)}, u^{(2)}$ be the second eigenvector of $L_A, L_W$ respectively.

We first show that the unperturbed $v^{(2)}$ can *clearly* distinguish the two outermost clusters and that $\lambda_1, \lambda_2$, and $\lambda_3$ (the first, second, and third smallest eigenvalues of $L_W$ respectively), are far away from each other. More precisely we show $|v_i^{(2)}| = \Theta(\frac{1}{\sqrt{n}})$ for all $i = 1, ..., n$ and its sign corresponds to the cluster identity of point $i$. Further the eigen-gap, $\lambda_2 - \lambda_1 = \lambda_2 = \Theta(n)$, and $\lambda_3 - \lambda_2 = \Theta(n)$. Now, using the well-known Davis-Kahan perturbation theorem, we can show that

$$||v^{(2)} - u^{(2)}||_2 = O\left(\sigma \frac{\sqrt{n \log n}}{\min(\lambda_2, \lambda_3 - \lambda_2)}\right) = O\left(\sigma \sqrt{\frac{\log n}{n}}\right)$$

The most straightforward way of turning this $l_2$-norm bound into uniform-entry-wise $l_\infty$ bound is to assume that only one coordinate has large perturbation and comprises all of the $l_2$-perturbation. We perform a much more careful analysis to show that all coordinates uniformly have low perturbation. Specifically, we show that if $\sigma = O(\sqrt[4]{\frac{\log n}{n}})$, then with high probability, $||v_i^{(2)} - u_i^{(2)}||_\infty = O(\sqrt{\frac{1}{n}})$.

Combining this and the fact that $|v_i^{(2)}| = \Theta(\frac{1}{\sqrt{n}})$, and performing careful comparison with the leading constants, we can conclude that spectral clustering will correctly recover the first split.

**Outline of proof of Theorem 2**

Leveraging our analysis of Theorem 1 we derive an $\ell_\infty$ bound on the bottom $k$-eigenvectors. One potential complication we need to resolve is that the $k$-Block Diagonal matrix has repeated eigenvalues and more careful *subspace* perturbation arguments are warranted.

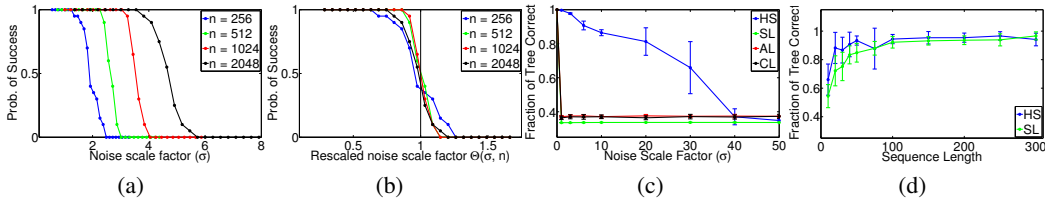

Figure 3: (a),(b): Threshold curves for the first split in HBMs. Comparison of clustering algorithms with $n = 512, m = 9$ (c), and on simulated phylogeny data (d).

We further propose a *different* algorithm, K-WAY SPECTRAL, from the standard $k$-means. The algorithm carefully chooses cluster centers and then simply assigns each point to its nearest center. The $\ell_\infty$ bound we derive is much stronger than $\ell_2$ bounds prevalent in the literature and in a straightforward way provides a no-error guarantee on K-WAY SPECTRAL.

**Outline of proof of Theorem 3**

As is typically the case with minimax analysis, we begin by restricting our attention to a small (but hard to distinguish) class of models, and follow this by the application of Fano's inequality. Models are indexed by $\Theta(n, \sigma, \gamma, I_1)$, where $I_1$ denotes the indices of the rows (and columns) in the first cluster. For simplicity, we'll focus only on models with $|I_1| = n/2$.

Since we are interested in the worst case we can make two further simplifications. The ideal (noiseless) matrix can be taken to be block-constant since the worst case is when the diagonal blocks are at their lower bound (which we call $p$) and the off diagonal blocks are at their upper bound ($q$). We consider matrices $W = A + R$, which are ($n \times n$) matrices, with $R_{ij} \sim \mathcal{N}(0, \sigma^2)$.

Given the true parameter $\theta_0$ we choose the following "hard" subset $\{\theta_1, \ldots, \theta_M\}$. We will select models which mis-cluster only the last object in $I_1$, there are exactly $n/2$ such models. Our proof is an application of Fano's inequality, using the Hamming distance and the KL-divergence between the true model $I_1$ and the estimated model $\hat{I}_1$. See the appendix for calculations and proof details.

The proof of Theorem 4 follows from a careful union bound argument to show that even amongst the combinatorially large number of balanced cuts of the graph, the true cut has the lowest weight.

## 5 Experiments

We evaluate our algorithms and theoretical guarantees on simulated matrices, synthetic phylogenies, and finally on two real biological datasets. Our experiments focus on the effect of noise on spectral clustering in comparison with agglomerative methods such as single, average, and complete linkage.

### 5.1 Threshold Behavior

One of our primary interests is to empirically validate the relation between the scale factor $\sigma$ and the sample size $n$ derived in our theorems. For a range of scale factors and noisy HBMs of varying size, we empirically compute the probability with which spectral clustering recovers the first split of the hierarchy. From the probability of success curves (Figure 3(a)), we can conclude that spectral clustering can tolerate noise that grows with the size of the clusters.

We further verify the dependence between $\sigma$ and $n$ for recovering the first split. For the first split we observe that when we rescale the x-axis of the curves in Figure 3(a) by $\sqrt{\log(n)/n}$ the curves line up for different $n$. This shows that empirically, at least for the first split, spectral clustering appears to achieve the minimax rate for the problem.

### 5.2 Simulations

We compare spectral clustering to several agglomerative methods on two forms of synthetic data: noisy HBMs and simulated phylogenetic data. In these simulations, we exploit knowledge of the true *reference tree* to quantitatively evaluate each algorithm's output as the fraction of triplets of leaves for which the most similar pair in the output tree matches that of the reference tree. One can verify that a tree has a score of 1 if and only if it is identical to the reference tree.

Initially, we explore how HS compares to agglomerative algorithms on large noisy HBMs. In Figure 3(c), we compare performance, as measured by the triplets metric, of four clustering algorithms (HS , and single, average, and complete linkage) with $n = 512$ and $m = 9$. We also evaluate

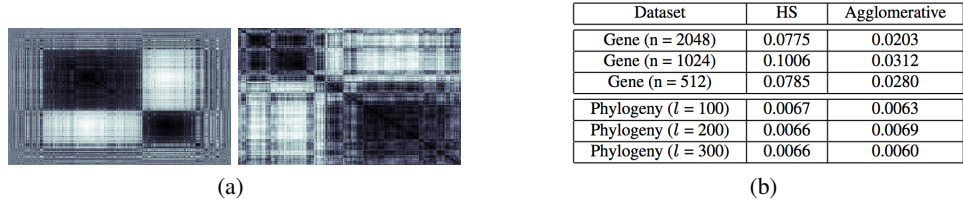

| Dataset | HS | Agglomerative |
|---|---|---|
| Gene (n = 2048) | 0.0775 | 0.0203 |
| Gene (n = 1024) | 0.1006 | 0.0312 |
| Gene (n = 512) | 0.0785 | 0.0280 |
| Phylogeny ($l$ = 100) | 0.0067 | 0.0063 |
| Phylogeny ($l$ = 200) | 0.0066 | 0.0069 |
| Phylogeny ($l$ = 300) | 0.0066 | 0.0060 |

(a)                                      (b)

Figure 4: Experiments with real world data. (a): Heatmaps of single linkage (left) and HS (right) on gene expression data with $n = 2048$. (b) $\Delta$-entropy scores on real world data sets.

HS and single linkage as applied to reconstructing phylogenetic trees from genetic sequences. In Figure 3(d), we plot accuracy, again measured using the triplets metric, of the two algorithms as a function of sequence length (for sequences generated from the `phyclust` R package [3]), which is inversely correlated with noise (i.e. short sequences amount to noisy similarities). From these experiments, it is clear that HS consistently outperforms agglomerative methods, with tremendous improvements in the high-noise setting where it recovers a significant amount of the tree structure while agglomerative methods do not.

### 5.3 Real-World Data

We apply hierarchical clustering methods to a yeast gene expression data set and one phylogenetic data set from the PFAM database [5]. To evaluate our methods, we use a $\Delta$-entropy metric defined as follows: Given a permutation $\pi$ and a similarity matrix $W$, we compute the rate of decay off of the diagonal as $\hat{s}_d \triangleq \frac{1}{n-d} \sum_{i=1}^{n-d} W_{\pi(i),\pi(i+d)}$, for $d \in \{1, ..., n-1\}$. Next, we compute the entropy $\hat{E}(\pi) \triangleq -\sum_{i=1}^{n-1} \hat{p}_\pi(i) \log \hat{p}_\pi(i)$ where $\hat{p}_\pi(i) \triangleq (\sum_{d=1}^{n} \hat{s}_d)^{-1} \hat{s}_i$. Finally, we compute $\Delta$-entropy as $\hat{E}_\Delta(\pi) = \hat{E}(\pi_{random}) - \hat{E}(\pi)$. A good clustering will have a large amount of the probability mass concentrated at a few of the $\hat{p}_\pi(i)$s, thus yielding a high $\hat{E}_\Delta(\pi)$. On the other hand, poor clusterings will specify a more uniform distribution and will have lower $\Delta$-entropy.

We first compare HS to single linkage on yeast gene expression data from DeRisi et al [4]. This dataset consists of 7 expression profiles, which we use to generate Pearson correlations that we use as similarities. We sampled gene subsets of size $n = 512, 1024$, and $2048$ and ran both algorithms on the reduced similarity matrix. We report $\Delta$-entropy scores in Table 4(b). These scores quantitatively demonstrate that HS outperfoms single linkage and additionally, we believe the clustering produced by HS (Figure 4(a)) is qualitatively better than that of single linkage.

Finally, we run HS on real phylogeny data, specifically, a subset of the PDZ domain (PFAM Id: PF00595). We consider this family because it is a highly-studied domain of evolutionarily well-represented protein binding motifs. Using alignments of varying length, we generated similarity matrices and computed $\Delta$-entropy of clusterings produced by both HS and Single Linkage. The results for three sequence lengths (Table 4(b)) show that HS and Single Linkage are comparable.

## 6 Discussion

In this paper we have presented a new analysis of spectral clustering in the presence of noise and established tight information theoretic upper and lower bounds. As our analysis of spectral clustering does not show that it is minimax-optimal it remains an open problem to further tighten, or establish the tightness of, our analysis, and to find a computationally efficient minimax procedure in the general case when similarities are not block constant. Identifying conditions under which one can guarantee correctness for other forms of spectral clustering is another interesting direction. Finally, our results apply only for binary hierarchical clusterings, yet $k$-way hierarchies are common in practice. A future challenge is to extend our results to $k$-way hierarchies.

## 7 Acknowledgements

This research is supported in part by AFOSR under grant FA9550-10-1-0382 and NSF under grant IIS-1116458. AK is supported in part by a NSF Graduate Research Fellowship. SB would like to thank Jaime Carbonell and Srivatsan Narayanan for several fruitful discussions.

## Footnotes

[1]Recall $a_n = o(b_n)$ and $b_n = \omega(a_n)$ if $lim_{n \to \infty} \frac{a_n}{b_n} = 0$

# References

[1] Dimitris Achlioptas and Frank Mcsherry. On spectral learning of mixtures of distributions. In *Computational Learning Theory*, pages 458–469, 2005.

[2] S. Charles Brubaker and Santosh Vempala. Isotropic pca and affine-invariant clustering. In *FOCS*, pages 551–560, 2008.

[3] Wei-Chen Chen. *Phylogenetic Clustering with R package phyclust*, 2010.

[4] Joseph L. DeRisi, Vishwanath R. Iyer, and Patrick O. Brown. Exploring the Metabolic and Genetic Control of Gene Expression on a Genomic Scale. *Science*, 278(5338):680–686, 1997.

[5] Robert D. Finn, Jaina Mistry, John Tate, Penny Coggill, Andreas Heger, Joanne E. Pollington, O. Luke Gavin, Prasad Gunesekaran, Goran Ceric, Kristoffer Forslund, Liisa Holm, Erik L. Sonnhammer, Sean R. Eddy, and Alex Bateman. The Pfam Protein Families Database. *Nucleic Acids Research*, 2010.

[6] Dorit S. Hochbaum and David B. Shmoys. A Best Possible Heuristic for the K-Center Problem. *Mathematics of Operations Research*, 10:180–184, 1985.

[7] Ling Huang, Donghui Yan, Michael I. Jordan, and Nina Taft. Spectral Clustering with Perturbed Data. In *Advances in Neural Inforation Processing Systems*, 2009.

[8] Ravindran Kannan, Hadi Salmasian, and Santosh Vempala. The spectral method for general mixture models. In *18th Annual Conference on Learning Theory (COLT*, pages 444–457, 2005.

[9] Frank McSherry. Spectral partitioning of random graphs. In *IEEE Symposium on Foundations of Computer Science*, page 529, 2001.

[10] Andrew Y. Ng, Michael I. Jordan, and Yair Weiss. On Spectral Clustering: Analysis and an Algorithm. In *Advances in Neural Information Processing Systems*, pages 849–856. MIT Press, 2001.

[11] Karl Rohe, Sourav Chatterjee, and Bin Yu. Spectral Clustering and the High-Dimensional Stochastic Block Model. *Technical Report 791, Statistics Department, UC Berkeley*, 2010.

[12] Sajama and Alon Orlitsky. Estimating and Computing Density Based Distance Metrics. In *ICML05, 22nd International Conference on Machine Learning*, 2005.

[13] Dan Spielman. *Lecture Notes on Spectral Graph Theory*, 2009.

[14] Terence Tao. Course notes on random Matrix Theory, 2010.

[15] Alexandre B. Tsybakov. *Introduction a lÉstimation Non-paramÃl'trique*. Springer, 2004.

[16] Ulrike von Luxburg. A Tutorial on Spectral Clustering. Technical Report 149, Max Planck Institute for Biological Cybernetics, August 2006.

[17] Ulrike von Luxburg, Mikhail Belkin, and Olivier Bousquet. Consistency of Spectral Clustering. In *The Annals of Statistics*, pages 857–864. MIT Press, 2004.

